# NeuralSteiner: Learning Steiner Tree for Overflow-avoiding Global Routing in Chip Design

**Ruizhi Liu**[1,2,4]     **Zhisheng Zeng**[1,2,5]     **Shizhe Ding**[1,2]     **Jingyan Sui**[1,2]
**Xingquan Li**[5,6]     **Dongbo Bu**[1,2,3]*

[1]SKLP, Institute of Computing Technology,
Chinese Academy of Sciences, Beijing 100190, China
{liuruizhi19s, dingshizhe19s, suijingyan18b, dbu}@ict.ac.cn
[2]University of Chinese Academy of Sciences, Beijing 101408, China
[3]Central China Artificial Intelligence Research Institute,
Henan Academy of Sciences, Zhengzhou 450046, Henan, China
[4]Beijing Institute of Open Source Chip, Beijing 100089, China
[5]Peng Cheng Laboratory, Shenzhen 518000, Guangdong, China
{zengzhsh, lixq01}@pcl.ac.cn
[6]School of Mathematics and Statistics,
Minnan Normal University, Zhangzhou 363000, Fujian, China

## Abstract

Global routing plays a critical role in modern chip design. The routing paths generated by global routers often form a rectilinear Steiner tree (RST). Recent advances from the machine learning community have shown the power of learning-based route generation; however, the yielded routing paths by the existing approaches often suffer from considerable overflow, thus greatly hindering their application in practice. We propose NeuralSteiner, an accurate approach to overflow-avoiding global routing in chip design. The key idea of NeuralSteiner approach is to learn Steiner trees: we first predict the locations of highly likely Steiner points by adopting a neural network considering full-net spatial and overflow information, then select appropriate points by running a graph-based post-processing algorithm, and finally connect these points with the input pins to yield overflow-avoiding RSTs. NeuralSteiner offers two advantages over previous learning-based models. First, by using the learning scheme, NeuralSteiner ensures the connectivity of generated routes while significantly reducing congestion. Second, NeuralSteiner can effectively scale to large nets and transfer to unseen chip designs without any modifications or fine-tuning. Extensive experiments over public large-scale benchmarks reveal that, compared with the state-of-the-art deep generative methods, NeuralSteiner achieves up to a 99.8% reduction in overflow while speeding up the generation and maintaining a slight wirelength loss within only 1.8%.

## 1   Introduction

In the modern design flow of Very Large Scale Integration (VLSI), global routing has become one of the most complex and time-consuming steps. Given the complexity of VLSI netlist [17] that contains millions or even billions of nets requiring routing, global routers must interconnect pins of nets, minimize the total wirelength of the routes while avoiding overflow (or congestion) in a strictly limited area of chip[2, 24]. Overflow occurs when the number of routes in a particular area of the

---

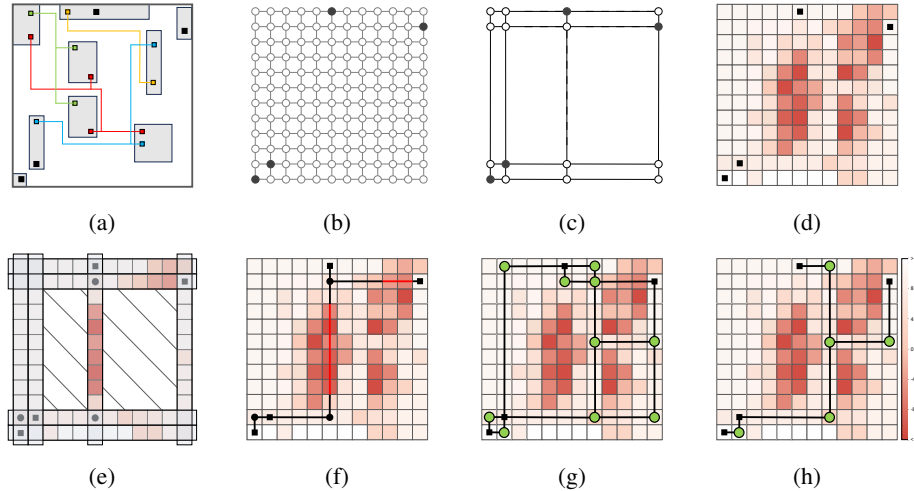

Figure 1: **An illustrative example to show overflow-avoiding global routing in chip design.** (a) Chip layout of a real-world net extracted from ISPD07_adaptec1. (b) Two-dimensional grid graph. (c) The Hanan grid of a 4-pin net. (d) The resource map and pin map (actually divided into two channels) obtained from (a) for this 4-pin net. (e) The predicted hub points (black circles) and stripe mask applied by Hubrouter [8]. (f) The routing result generated by HubRouter [8] suffers from congestion (red edges). (g) The candidate points (green circles) predicted by NeuralSteiner and the corresponding net augmented graph (NAG). (h) The final routing result generated by NeuralSteiner that avoids overflow.

chip exceeds the available routing resources or routes cross through impassable obstacles, which significantly impacts the subsequent design flow and functionality realization of the chip [24, 22]. Even the two-pin routing under design constraints or obstacles turns out to be NP-complete. When the number of pins exceeds two, the routing problem can often be transformed into the construction problem of rectilinear Steiner minimum tree (RSMT) [10], which is also NP-complete and becomes even more challenging when considering avoiding overflow [22, 16, 24, 6].

Traditional global routers propose various human-designed heuristics to obtain near-optimal solutions for RSMT [7, 27, 13, 16, 24] or directly solve the integer programming problem for concurrent routing of multiple nets [6, 33]. Recent advances in applying learning-based methods to chip design problems have shown feasibility and powerful abilities and even surpassing the performance of human expert-designed algorithms, such as using deep reinforcement learning (DRL) for placement [29, 18, 19] and using convolutional neural network (CNN) for predicting design rule violations [34]. In global routing, DRL is firstly adopted to explore surrounding directions of current positions and achieve successful connectivity on small-scale nets [20], while REST [23] decomposes multi-pin net into 2-pin pairs and explores the sequence of pin pairs through DRL to form the RST. Moreover, recent works adopt deep generative models [32, 4] to perform one-shot generation of nets. To address the connectivity problems of generative methods, HubRouter[8] decomposes global routing generation into hub-generation phase and pin-hub-connection phase to sequentially connect the pin-hub pairs, thereby successfully ensuring the connectivity of the routes.

However, current learning-based routing methods suffer from high overflow within their routing results, primarily due to inadequate consideration of resource availability during the routing process. Existing DRL-based approaches [20, 23, 28] tend to solely focus on wirelength as the reward during action space exploration, while other generative methods do incorporate the current routing area's resource status in their inputs but mainly aim for connectivity or wirelength optimization in their post-processing phases [4, 32, 5, 8]. Consequently, our experimental findings indicate that routes generated by HubRouter still exhibit significant congestion, which is illustrated in Fig. 1f.

To address these challenges, we propose a congestion-aware learning scheme named NeuralSteiner, which consists of two main phases: $i$) **Candidate point prediction phase**: Utilizing a neural network combined with full-image spatial and overflow information aggregation to predict the accurate locations of what we call candidate points for overflow-avoiding rectilinear Steiner tree; $ii$) **Overflow-**

**avoiding RST construction phase**: Constructing an augmented graph of the net based on the predicted candidate points and calculating the overflow-avoiding RST using a simple but effective greedy algorithm. Through this two-phase setup, NeuralSteiner successfully ensures connectivity and enables the generation of overflow-avoiding routing results for large-scale nets.

This paper has **three main contributions**:

- We propose NeuralSteiner, a two-phase global routing scheme, which to our knowledge, is the first learning-based approach capable of optimizing both wirelength and overflow and effectively addressing the routing problem of large-scale nets.

- We devise a neural network architecture that integrates the deep residual network with recurrent crisscross attention mechanism to learn the Steiner point locations from a carefully curated expert dataset and propose a post-processing algorithm based on augmented graphs to construct routes with substantially less overflow than recent works.

- We conduct extensive experiments on 14 public large-scale routing benchmarks compared with the state-of-the-art learning-based method, where the NeuralSteiner achieves up to 99.6% reduction in total overflow with a wirelength loss within 1.8%. Moreover, Neural-Steiner can generate overflow-avoiding routes for nets with more than 1000 pins, previously challenging for recent works, which narrows the gap between learning-based methods and practical routing applications.

## 2   Preliminaries and Related Works

**Global Routing.** Given the complexity of VLSI routing problems, the circuit layout like 1a is partitioned into rectangular areas known as global cells (GCells) [6]. The global routing problem can be modeled as a grid graph $G(V, E)$, where each GCell is represented as a vertex ($v \in V$), and adjacent GCells are connected by an edge ($e \in E$) that represents the boundary between GCells. Chip designs often contain two or more metal layers for routing. Each metal layer is dedicated to either horizontal or vertical direction and the projection of these layers onto a two-dimensional grid graph is shown in Fig.1b. Global router will assign a set of GCells interconnected by numerous edges to each net as its routing result to connect all pins, which often forms a Rectilinear Steiner Tree (RST) [7]. The concepts of Hanan grid [11] and escape graph [9] are often used for the generation of the shortest RSMT avoiding obstacles [22], considering the intersection points in these graphs as candidate locations for Steiner points. However, due to the complex and irregular distribution of congestion, the construction of escape graph becomes complicated, while the Hanan grid is ineffective at circumventing congestion, which is shown in Fig.1c.

**Overflow.** Give edge $e(u, v) \in E$ is the boundary between GCell $u$ and GCell $v$, the capacity $c(u, v)$ is the routing resource of edge $e$ that can be provided to global router and demand $d(u, v)$ is the number of routes passing through edge $e$. The resource $r(u, v)$ of edge $e$ is the part of the capacity that can still be utilized to route, which is defined in Equation (1):

$$r(u, v) = c(u, v) - d(u, v) \tag{1}$$

Overflow occurs when $r(u, v) < 0$. The routing results containing overflow generated by the global router will not be accepted by subsequent routing process and will trigger a time-consuming rip-up and reroute iteration in order to eliminate overflow [2]. Therefore, the global router should not only attempt to find the shortest connection for each net but also minimize the number of overflow.

**Traditional Global Router.** Traditional routing algorithms typically divide global routing into two main stages to address congestion: Steiner topology generation and rip-up and reroute (RRR). The former utilizes the FLUTE algorithm [7], based on lookup tables, to generate Steiner trees with nearly minimal wirelength for each net. However, FLUTE is unaware of congestion. During this phase, most routers only use edge shifting to partially mitigate congestion by moving some edges out of congested areas [7], while CUGR-2 [25] applies the construction of augmented graphs to build candidate paths for nets' RSTs, adjusting the position of certain Steiner points to circumvent potential congestion. In order to resolve congestion in the RSTs, traditional routers will invoke RRR, iteratively removing all initially routed nets in congested areas and employing maze routing that optimizes wirelength and congestion simultaneously. This process becomes dramatically time-consuming as the chip design's scale and complexity rise. Hence, accelerating congestion resolution through deep learning-based methods can enhance the overall performance of global routing algorithm.

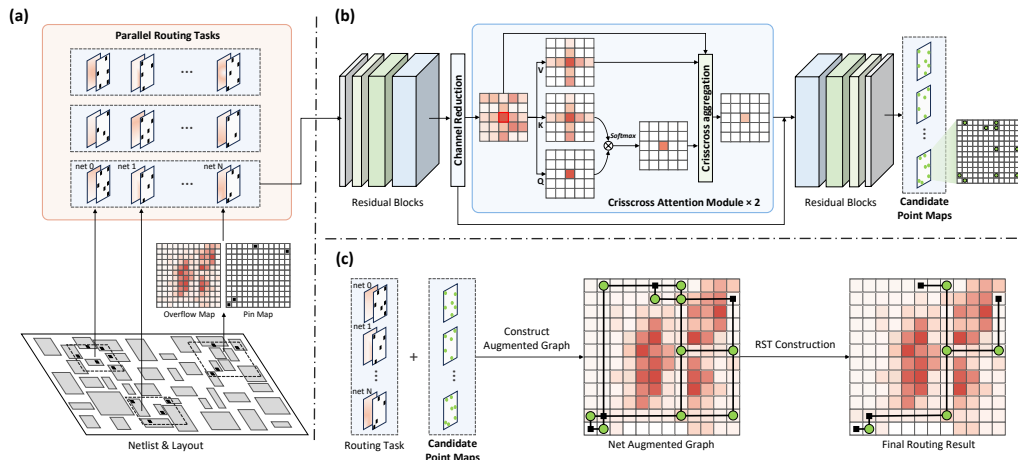

Figure 2: **Overview of NeuralSteiner.** (a) The parallel routing tasks accelerate routing by grouping the non-overlapping nets into one batch. (b) During the first phase, NeuralSteiner predicts the candidate point locations for the RST with full-image aggregation of spatial and overflow information. (c) During the second phase, NeuralSteiner constructs the net augmented graphs based on the predicted candidate points and generates overflow-avoiding RSTs.

**Learning-based RST construction.** Various works explore the feasibility and advantages in wirelength and efficiency of applying deep neural networks to global routing, including generation of the pin-connection order [23], segments [4, 32] or custom hub points of RST [8]. However, most of the challenges in actual global routing come from the complexity of large-scale nets and how to avoiding overflow when routing resources are limited. Under the circumstances, detours are indispensable to get rid of congestions, while the shortest RST like Fig. 1f generated by HubRouter [8] is not practically usable. The chip layout can be viewed as an image, where each pixel represents a tile in global routing, and images of different channels represent the locations of the pins and capacity of the grid edges. The output points can also be represented as a binary image. But unlike the four kinds of hub points defined in Hubrouter, we simplify the learning target in RST construction and select Steiner points and corner points in RST as candidate points to learn. Formally, we have

**Definition 1.** *Candidate point Given an $m \times n$ binary image representing the RST, where each pixel $p_{xy}(1 \le x \le m, 1 \le y \le n)$ represents whether the position is occupied by a route and $p_{0y} = p_{(m+1)y} = p_{x0} = p_{x(n+1)} = 0$. The pixel $p_{xy}$ is a candidate point if and only if it satisfies:*

$$\begin{cases} p_{xy} = 1, & (d_{xy} > 2) \\ p_{xy} = 1, \text{ and } p_{(x-1)y} + p_{(x+1)y} = 1, \text{ and } p_{x(y-1)} + p_{x(y+1)} = 1, & (d_{xy} = 2) \end{cases} \quad (2)$$

*where $d_{xy} = p_{(x-1)y} + p_{(x+1)y} + p_{x(y-1)} + p_{x(y+1)}$ denotes the degree of this point in the RST.*

Using Definition. 1, the Steiner points and corner points can be recognized as candidate points in the pixel image of RST. The differences between hub point in [8] and candidate point are visualized in Fig. S1.

## 3 Method

### 3.1 Overall Pipeline

NeuralSteiner decomposes the global routing process into two main phases to optimizes wirelength and overflow of the routing result simultaneously. Before introducing the main methods, we first propose our parallel task construction in Sec. 3.2. We then introduce the candidate point prediction method with aggregation of full-scale spatial and overflow information in Sec. 3.3. An augmented graph-based overflow-avoiding RST construction method will be proposed in Sec. 3.4. The overall pipeline of NeuralSteiner is illustrated in Fig.2.

## 3.2 Parallel Routing Tasks Construction.

Routing of two nets cannot be parallelized if the bounding boxes of their pins have overlap, which is defined as conflict between two nets. Inspired by [26], we scan and group the non-conflicting nets into a set $t$, which is called a routing task, which divides the numerous nets in the design into a set of mutually conflicting routing tasks. Nets within a task $t$ can be batched together and fed into the neural network for prediction and post-processing, significantly enhancing the parallelism of routing. Please refer to App. B.3 for the detail of our routing tasks construction algorithm.

## 3.3 Candidate Point Prediction

Candidate points prediction can be formulated as an image segmentation task [31], which involves training a neural network model to perform pixel-level classification to recognize the locations of candidate points found in the expert RSTs data. We will first introduce the expert routes dataset optimized for both wirelength and overflow, then the network architecture incorporating the recurrent crisscross attention module to tackle the complex large-scale nets, as well as the design of our training protocol.

**Expert Routing Dataset Construction.** We utilize a state-of-the-art traditional global router named CUGR [24] to perform routing on public benchmarks [30] also used by [8] and extract the overflow map and pin map of every net in real-time. We adopt the logistic function in CUGR to calculate the overflow value using resource $r(u, v)$:

$$lg(u,v) = (1.0 + exp(slope \times r(u,v)))^{-1}. \tag{3}$$

where $slope$ (which is set to 1 here) is an adjustable parameter that determines the global router's sensitivity to overflow and the overflow value will increase rapidly as the resources are being used up. After that, we directly employ CUGR's maze routing algorithm to execute the rip-up and reroute process to obtain the congestion-avoiding routing results. We mark the Steiner points and corner points in the RSTs constructed by CUGR as candidate points and generate the label candidate point map for every net. Rather than clipping all images to the same scale $64 \times 64$, which is set in HubRouter [8], we maintain three maps of every net at the original scale of its bounding box. This preserves the precise spatial and overflow information and does not exclude any large-scale nets.

**Network Architecture.** In order to tackle the problem of large variation in net scale, we employ a ResNet structure as the backbone of our model and combined it with the recurrent crisscross attention mechanism [14] to encoding full-net overflow and long-range associations in the input features. Convolutional neural networks (CNN) have been proven to be efficiently applied in chip design like predicting chip congestion distribution [35], DRV distribution [34], and thermal distribution [3]. However, due to the fixed geometric structures, CNN is inherently limited to local receptive fields that face difficulties in capturing long-range correlations. Thus, we introduce the recurrent crisscross attention mechanism (RCCA) to aggregate features from all pixels on the feature map. We insert one RCCA module with two e crisscross attention blocks in ResNet. Fig.2 illustrates the network architecture of NeuralSteiner. We also remove the down-sampling operations to retain more spatial details of feature maps because the construction of RST requires accurate spatial location information when connecting pins that are far apart in large-scale nets. Through the computation of RCCA, the network can aggregate information of pins and congestion over the whole scale of feature maps, thereby enhancing the quality of candidate points prediction for large RSTs. This will be further demonstrated in ablation study in Sec. 4.4. The implementation details of our network and RCCA calculation are shown in App. B.1.

**Model Training.** We adopt focal loss [21] $\ell_{focal}$ to mitigate the imbalance between positive and negative class samples in training data where the candidate points in RST only occupy a minority of pixels in the entire routing area. Let $p_t$ be the predicted probability for the ground truth class $t$, $\ell_{focal}$ is defined as:

$$\ell_{focal} = -\alpha_t(1 - p_t)^\gamma \log(p_t) \tag{4}$$

where $\alpha_t$ is the weighting factor while $\gamma$ is the focusing parameter that reduces the loss for well-classified examples. We also adopt the dice loss $\ell_{dice}$ to measure the similarities between the predicted candidate points and the ground truth. Using $p_{xy}$ to represent the probability of pixel at position $(x, y)$ predicted as a candidate point and $g_{xy}$ to represent the label, $\ell_{dice}$ can be expressed as:

$$\ell_{dice} = 1 - \frac{2\sum_{x,y} p_{xy}g_{xy} + \epsilon}{\sum_{x,y} p_{xy} + \sum_{x,y} g_{xy} + \epsilon} \tag{5}$$

where $\epsilon$ is a small constant added to avoid division by zero. Additionally, since the global routing problem is NP-complete, even expert router may not generate the optimal routing solution for the net that achieves the shortest wirelength with the minimal overflow. Therefore, we further add an overflow loss $\ell_{of}$ to measure the congestion status of predicted points. Let $o_{xy}$ be the value at position $(x, y)$ of the overflow map, $\ell_{of}$ can be calculated by:

$$\ell_{of} = \frac{\sum_{x,y} p_{xy} o_{xy} + \epsilon}{\sum_{x,y} p_{xy} + \epsilon} \tag{6}$$

The inclusion of overflow loss helps the model identify potential candidate points that are not in the label set but have a lower intrinsic congestion, benefiting the post-processing algorithm for overflow-avoiding RST construction. Then the trainable model $\theta$ is determined at the training stage by minimizing the loss function as follows:

$$\mathcal{L}(\theta) = c_{fl} \cdot \ell_{focal} + c_{di} \cdot \ell_{dice} + c_{of} \cdot \ell_{of} \tag{7}$$

where $c_{fl}, c_{di}, c_{of}$ represent weight of corresponding loss item. The parameters used in training process are provided in App. B.2.

### 3.4 Overflow-avoiding RST Construction

To construct an overflow-avoiding Rectilinear Steiner Tree (RST) based on the candidate points predicted by the neural network, we will first introduce the construction of net augmented graph that contains potential overflow-free edges and then propose a simple and effective greedy RST construction algorithm. Unlike previous works that focus solely on minimizing wirelength, the inclusion of the irregular distribution of congestion makes solving for an RST more challenging.

**Net Augmented Graph.** We introduce the concept of the net augmented graph (NAG) based on neural network-predicted points to avoid congestion. We first merge the predicted candidate point map and pin map, then sequentially examine each point $p_{xy} \geq 1$ from the merge map according to the following two conditions: 1) if this point shares the same horizontal (X) or vertical (Y) coordinates with another point $q$, and 2) if there is no other points on the line connecting $p$ and $q$. Then an edge $e(p, q)$ will be established if the above two conditions are met and the weight of $e(p, q)$ is set as

$$\mathcal{W}(e) = w_d(|x_p - x_q| + |y_p - y_q|) + w_o \sum_{x,y} o_{xy} \tag{8}$$

where $min(x_p, x_q) \leq x \leq max(x_p, x_q), min(y_p, y_q) \leq y \leq max(y_p, y_q)$. $\mathcal{W}(e)$ balances the wirelength and congestion of the edge by using weights $w_d = 1.0$ and $w_o = 5.0$. After examining all the points, to ensure the connectivity of the net, we will check the connectivity of the current NAG and add candidate point and edge between different connected components if this NAG is disconnected. App. B.4 provides a detailed introduction to the construction algorithm.

Note that in HubRouter [8], stripe mask is introduced as a filter that removes noise hub points to limit the solution space similar to the Hanan grid, which ensure that the wirelength as short as possible. However, as dipicted in Fig.1e, the addition of stripe mask in HubRouter limits its ability to generate RST avoiding congested areas. On the contrary, we here retain all candidate points predicted by the model and constructed the NAG based on them, which reduces the complexity of solving RST while preserving the solution space to avoid overflow.

**Overflow-avoiding RST Construction.** We convert the calculation of the overflow-avoiding RST into a greedy construction of minimal spanning tree that connects all pins. Initially, we consider all pins as separate connected components containing only one node. In each iteration, based on the NAG, we greedily select and connect the path between the two nearest connected components, then update the shortest distance (the sum of the weights of all edges on the path) of the newly formed connected component to all other connected components. This operation repeats until all pins are included in one connected component. Since this method may generate additional detours, we use a simple algorithm to detect potential feasible path reuse to shorten the wirelength. Furthermore, to accelerate the construction of RST, we parallelize the computation of the shortest distances between pins or connected components on the NAG. For the detailed algorithm and analysis of time complexity and scalability, please refer to App. B.5.

# 4    Results and Discussion

## 4.1    Datasets and Experiment Setting

For training, we construct the training set from ISPD07[30] using the method described in Sec. 3.3. Since our network's input size is variable, we limit the nets' Half-perimeter wirelength (HPWL) in the training set to $HPWL \leq 128$, instead of fixing both width and height to 64. For test, in Sec. 4.2 we use the same settings from HubRouter [8] to divide samples outside the training set into four groups of small-scale nets to compare the connectivity and wirelength of NeuralSteiner and HubRouter. For more extensive experiments, in Sec. 4.3 we select six public chip designs (ibm01-06) from ISPD98 [1] and eight two-layer large-scale chip designs (adaptec(01-05)_2d, newblue(01-03)_2d) from ISPD07 (with no overlap with the training set) to perform global routing on all nets in these designs, comparing total overflow, wirelength and generation time. The ablation and generalization studies for NeuralSteiner are also conducted on chip designs from ISPD07. We repeat 3 times under different seeds for HubRouter on the small nets test set and ISPD98, and then choose the seed with best overflow for HubRouter (GAN) to conduct the ISPD07 experiment. More details about the experimental benchmark information and hyperparameter settings can be found in App. B.2.

## 4.2    Connectivity and Wirelength on Small Nets

We compare NeuralSteiner with three different architectures of HubRouter on the same test set from part of ISPD07 benchmarks, which is divided into 'Route-small-4', 'Route-small', 'Route-large-4' and 'Route-large'. The number '4' in their names represents no more than or more than 4 pins, while 'small' and 'large' represent whether the Half-perimeter wirelength (HPWL) of the net is less or more than 16. The size of all nets' input map is fixed at $64 \times 64$. We do not include PRNet [4] as it shows very poor connectivity on 'large' net in previous work [8]. As shown in Table S2, NeuralSteiner ensures connectivity on this small-scale net test set, while achieving a wirelength rate (WLR) comparable to HubRouter. Due to the presence of recurrent crisscross attention calculation, our method is slightly behind in generation time.

## 4.3    Global Routing on Large-scale Benchmarks

To conduct extensive experiments, we first compare the proposed NeuralSteiner with three versions of HubRouter [8] and traditional global routers Boxrouter [6], GeoSteiner [15] and FLUTE + Edge Shifting [7] on ibm01-06 benchmarks from ISPD98. We then conduct fully routing of 8 chip designs from ISPD07 using GeoSteiner, FLUTE + Edge Shifting, HubRouter and our method. Note that in our experiments, we do not use the randomly generated nets from previous works [20], as they are relatively simple and have no overflow in the results. Moreover, it has been already studied in HubRouter that the DQN method takes excessively long time to run on even very small cases and PRNet [4] also lags behind HubRouter in terms of wirelength, time and overflow, so they are not included in the comparison.

Table 1: **Wirelength (WL) and running time on ISPD-98 (ibm01-06).** NeuralSteiner is compared with 2 traditional baselines and HubRouter with 3 generative structures (HR-VAE, HR-DPM, HR-GAN). Optimal results of WL and time are in bold.

| Metric | Model | ibm01 | ibm02 | ibm03 | ibm04 | ibm05 | ibm06 |
|---|---|---|---|---|---|---|---|
| WL | GeoSteiner | **60142** | **165863** | **145678** | **162734** | **409709** | **275868** |
| | Boxrouter | 62659 | 171110 | 146634 | 167275 | 410614 | 277913 |
| | FLUTE+ES | 61492 | 169251 | 146287 | 167547 | 411936 | 280477 |
| | HR-VAE | $64812 \pm 1252$ | $176838 \pm 6419$ | $161032 \pm 3231$ | $179018 \pm 4791$ | $440302 \pm 4577$ | $301035 \pm 5836$ |
| | HR-DPM | $66575 \pm 1394$ | $190142 \pm 2511$ | $168550 \pm 2103$ | $183051 \pm 1946$ | $474463 \pm 6674$ | $320423 \pm 2958$ |
| | HR-GAN | $60971 \pm 290$ | $167316 \pm 578$ | $146893 \pm 315$ | $164084 \pm 299$ | $411887 \pm 4529$ | $277977 \pm 514$ |
| | NeuralSteiner | 61735 | 170405 | 148036 | 166648 | 415684 | 283727 |
| Time (Sec) | GeoSteiner | **1.00** | **2.21** | **1.68** | **2.19** | **3.69** | **3.38** |
| | Boxrouter | 5.33 | 9.76 | 8.42 | 31.69 | 10.75 | 24.94 |
| | FLUTE+ES | 2.90 | 4.71 | 5.87 | 17.16 | 6.83 | 13.64 |
| | HR-VAE | $8.41 \pm 0.03$ | $8.47 \pm 0.06$ | $8.59 \pm 0.04$ | $10.85 \pm 0.04$ | $12.44 \pm 0.18$ | $15.83 \pm 0.11$ |
| | HR-DPM | $1701.57 \pm 34.19$ | $2589.93 \pm 19.63$ | $2669.28 \pm 22.77$ | $3593.04 \pm 24.10$ | $3995.47 \pm 19.57$ | $4305.82 \pm 132.85$ |
| | HR-GAN | $37.40 \pm 0.37$ | $41.55 \pm 0.51$ | $50.84 \pm 2.84$ | $59.94 \pm 2.75$ | $69.42 \pm 4.03$ | $81.96 \pm 3.98$ |
| | NeuralSteiner | 27.18 | 34.79 | 46.24 | 50.37 | 75.99 | 70.32 |

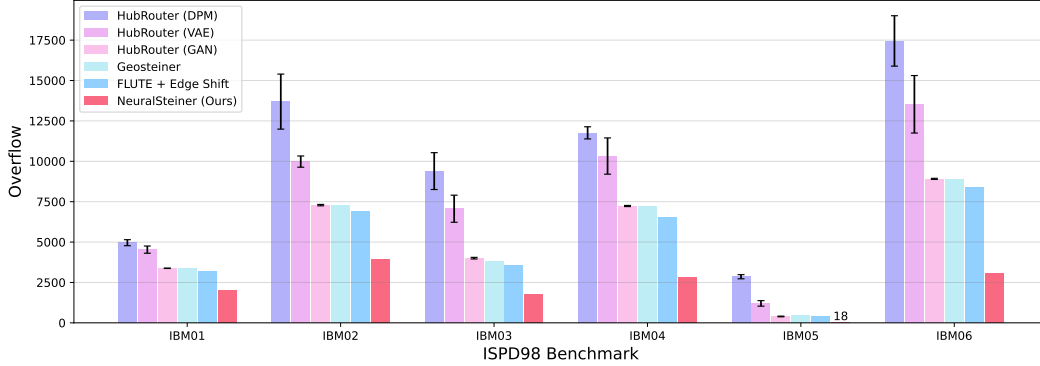

Figure 3: **Overflow on ISPD98 (ibm01-06).** Overflow of Geosteiner, HubRouter (VAE, DPM, GAN) and NeuralSteiner on ISPD-98 (ibm01-06) cases. Note that NeuralSteiner causes only 18 overflows on ibm05, which is annotated in the figure.

Table 2: **Evaluating NeuralSteiner and comparing it with state-of-the-art approaches on ISPD-07 (adaptec(01-05)_2d, newblue(01-03)_2d).** Overflow (OF), wirelength (WL) and running time are compared among traditional router GeoSteiner, FLUTE + Edge Shift and HubRouter with GAN structures (HR-GAN), which achieves the best overflow and wirelength among three kinds of HubRouters on ISPD98. Optimal results of overflow, wirelength and time are in bold.

| Metric | Method | adaptec01_2d | adaptec02_2d | adaptec03_2d | adaptec04_2d | adaptec05_2d | newblue01_2d | newblue02_2d | newblue03_2d |
|---|---|---|---|---|---|---|---|---|---|
| OF | GeoSteiner | 35945 | 53848 | 142254 | 45050 | 102300 | 1734 | 1832 | 584761 |
| | FLUTE+ES | 32518 | 50947 | 137104 | 42306 | 957704 | 1348 | 1713 | 558047 |
| | HR-GAN | 35441 | 53652 | 142131 | 45230 | 102108 | 1516 | 1857 | 583901 |
| | NeuralSteiner | **82** | **255** | **728** | **97** | **431** | **5** | **35** | **10343** |
| WL | GeoSteiner | **3389601** | **3209172** | **9330748** | **8865643** | **9784471** | **2320456** | **4595235** | **7371273** |
| | FLUTE+ES | 3418461 | 3235803 | 9417934 | 8896007 | 9886249 | 2347941 | 4651033 | 7454720 |
| | HR-GAN | 3407033 | 3229110 | 9355980 | 8888775 | 9832110 | 2339204 | 4623006 | 7391055 |
| | NeuralSteiner | 3438717 | 3247429 | 9459117 | 9003952 | 9915795 | 2365499 | 4668079 | 7480679 |
| Time (Sec) | GeoSteiner | **83.17** | **111.92** | **320.08** | **267.13** | **261.43** | **124.68** | **183.82** | **315.48** |
| | FLUTE+ES | 118.48 | 187.03 | 396.51 | 376.72 | 360.68 | 169.36 | 223.55 | 438.79 |
| | HR-GAN | 593.02 | 780.44 | 1324.81 | 1387.01 | 1384.96 | 849.34 | 1221.16 | 1526.86 |
| | NeuralSteiner | 347.20 | 461.35 | 1351.91 | 1138.66 | 1106.54 | 390.34 | 446.68 | 1225.79 |

**Routing Results on ISPD98.** Table 1 shows the total wirelength and generation time for all methods on ISPD98 benchmark. Since the total overflow of the traditional router Boxrouter is 0, we depict the routing overflow of the other methods in Fig. 3. NeuralSteiner significantly reduces the total overflow compared to the state-of-the-art learning-based method HubRouter (GAN), with an average reduction of 61.1% and up to 95% on ibm05. In terms of wirelength, NeuralSteiner does not incur much additional loss, maintaining it within 1.8%. Furthermore, due to the construction of the net parallel routing tasks, NeuralSteiner achieves shorter generation time compared with HubRouter (GAN). The comparison of the actual solutions between NeuralSteiner and Hubrouter is given in Fig. S2.

**Routing Results on ISPD07.** Based on the experimental results on ISPD98, we select four methods GeoSteiner, HubRouter (HR-GAN), and NeuralSteiner for comparison on the larger-scale ISPD07 chip designs. The summary of ISPD07 benchmarks we use is detailed introduced in Table S1, as well as the number of predicted candidate points for ISPD07. According to Table S1, the average number of candidate points added by NeuralSteiner is not significantly more than the average number of pins, which means that for the vast majority of nets, the number of nodes in the NAG will remain at a small scale and keep friendly to the calculation of the overflow-avoiding RST algorithm introduced in Sec. 3.4. The total overflow (OF), wirelength (WL) and generation time are shown in Table 2. According to Table 2, as the sizes of chip designs and nets further increase, NeuralSteiner achieves more dramatic reduction in total overflow, with an average reduction of 97.8% across all eight designs, and up to a 99.8% reduction on design adaptec04_2d, while the increase in wirelength still remains within 1.8% compared to HubRouter.

## 4.4 Generalization and Ablation Study

**Generalization.** Note that the NeuralSteiner proposed by this work is trained on small-scale nets with HPWL less than 128 and is tested on 14 large-scale public benchmarks, which contain net with HPWL even larger than 2000. Its ability to generalize to unseen and large-scale nets can be demonstrated. Fig. 4 shows the remaining resource map of adaptec01_2d design after routing by HubRouter and the proposed NeuralSteiner respectively, which demonstrates that NeuralSteiner can generalize to larger chips by using routing resource more evenly and avoiding the vast majority of overflow. Moreover, to extensively examine the ability of NeuralSteiner to mitigate overflow in the post-routing results, we integrate our method into CUGR and compare it with the original CUGR on post-detailed routing metrics on ISPD18/19 benchmarks, which are much larger than ISPD98 and ISPD07. The detailed routing is conducted by a commonly used detailed router DRCU. Short and space are two kinds of design rule violation caused by overflow in the detailed routing process. Table 3 shows that by integrating NeuralSteiner into CUGR, we achieve 4.4% and 19.1% reduction on average in shorts and spaces respectively, with minimal losses in wirelength and vias. This demonstrates that NeuralSteiner, as a pre-routing overflow mitigation method, is beneficial for reducing overflow in the post-routing results.

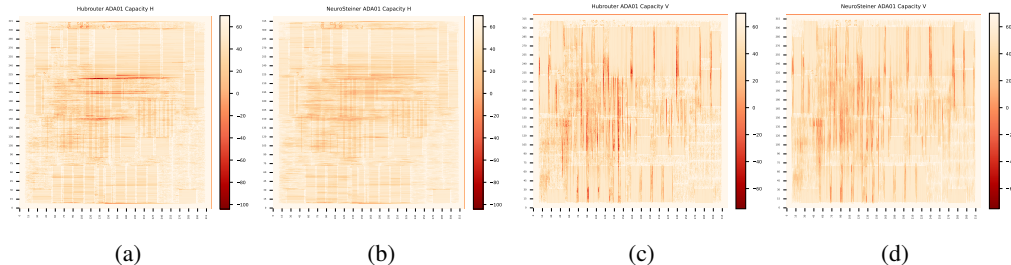

| (a) | (b) | (c) | (d) |

Figure 4: **The Overflow Distribution after routing by HubRouter and NeuralSteiner. (a), (c):** the horizontal and vertical overflow of HubRouter; **(b), (d)**: the horizontal and vertical overflow of NeuralSteiner. Depth of red color indicates the number of overflow.

Table 3: **Comparison of CUGR + NeuralSteiner and original CUGR on post-detailed routing metrics on ISPD18/19 benchmarks.** The detailed routing is conducted by DRCU. Short and space are two kinds of design rule violation caused by overflow.

| Design | Wire Length ($\times 10^7$) | | Via ($\times 10^5$) | | Short | | Space | |
| :---: | :---: | :---: | :---: | :---: | :---: | :---: | :---: | :---: |
| | CUGR | CUGR+NS | CUGR | CUGR+NS | CUGR | CUGR+NS | CUGR | CUGR+NS |
| ispd18_t5m5 | 2.878 | **2.874** | **9.154** | 9.180 | 389.5 | **362.5** | 16 | **7** |
| ispd18_t8m5 | **6.653** | 6.671 | 22.466 | **22.452** | 414.6 | **412.8** | 66 | **65** |
| ispd19_t7 | **12.556** | 12.621 | **40.446** | 40.956 | 2117.6 | **2042.3** | 7084 | **6472** |
| ispd19_t7m5 | **11.273** | 11.303 | **40.356** | 40.613 | 2368.5 | **2219.0** | 7715 | **6964** |
| Average | **1** | 1.002 | **1** | 1.005 | 1 | **0.956** | 1 | **0.809** |

**Ablation Study.** To study the role of recurrent crisscross attention (RCCA) module, as well as our loss function, we respectively ablate the RCCA module from the network architecture and the overflow loss $\ell_{of}$ from the loss function and keep other training settings the same. The modified models are tested in comparison with the unchanged NeuralSteiner on ibm01 and adaptec05_2d. Furthermore, to validate the effectiveness of our two-phase method, especially the effectiveness of predicted candidate points in reducing overflow, we ablate the output of the neural network and compare it with the unchanged NeuralSteiner. Results in Table 4 indicate that, although the wirelength of three modified models are not significantly affected due to graph-based post-processing, there are notable increases in overflow, especially on larger adaptec05_2d design. This implies that both RCCA module and overflow loss can help NeuralSteiner learn congestion-avoiding candidate points and acquire a better generalization ability to larger nets. Additionally, relying solely on the graph-based RST construction to generate RST on the Hanan grid without predicted points leads to more than $20\times$ increase in overflow on adaptec05_2d. This demonstrates that the neural network in our NeuralSteiner has learned an distribution of better candidate points for overflow-avoiding RST under tight resource

constraints. We also study the role of our post-processing by replacing the REST method and the stripe mask used in the hub-pin-connection phase of HubRouter by our NAG-based RST construction algorithm. As shown in Table 5, although the congestion of this combination is still $11\times$ larger than that of NeuralSteiner, it has decreased by nearly 95% compared to the original HubRouter, which fully demonstrates the effectiveness of our post-processing method.

Table 4: **Ablation study of different components in NeuralSteiner.** Comparison of the complete NeuralSteiner learning scheme with models removing neural networks or overflow loss $\ell_{of}$ or RCCA module.

| Metric | Model | ibm01 | adaptec05_2d |
|--------|-------|-------|--------------|
| OF | Without NN | 3015 | 10905 |
| | Without $\ell_{of}$ | 2258 | 2795 |
| | Without RCCA | 2189 | 2290 |
| | NeuralSteiner | **2033** | **431** |
| WL | Without NN | 62108 | 9931120 |
| | Without $\ell_{of}$ | 61965 | 9927082 |
| | Without RCCA | 62098 | 9930587 |
| | NeuralSteiner | **61735** | **9915795** |

Table 5: **Ablation study of NAG-based RST construction.** Comparison of original HubRouter (GAN), HubRouter (GAN) with our NAG-based RST construction without stripe mask and NeuralSteiner. Optimal results of overflow and wirelength are in bold.

| Metric | Model | adaptec05_2d |
|--------|-------|--------------|
| OF | HR-GAN | 102108 |
| | HR-GAN(w/o Mask)+NAG | 5245 |
| | NeuralSteiner | **431** |
| WL | HR-GAN | **9832110** |
| | HR-GAN(w/o Mask)+NAG | 9932476 |
| | NeuralSteiner | 9915795 |

## 5 Conclusion

In this paper, we introduce NeuralSteiner, a two-phase learning-based global routing scheme. By combining neural network-predicted candidate points with a post-processing method based on net augmented graph, NeuralSteiner can generate overflow-avoiding and connectivity-assured routing solutions for unseen large-scale nets in one shot, substantially reducing the overflow by up to 99.8% on real-world chip benchmarks, which narrows the gap between learning-based routing method and practical chip routing applications.

The main limitation of our method is that we still rely on heuristic post-processing algorithms for Rectilinear Steiner Tree (RST) construction, which leads to time-consuming calculations and a slight increase in wirelength. In the future, we will explore using continuous probabilistic candidate point maps and investigate end-to-end learning with neural networks for generating overflow-avoiding RSTs. Addressing this limitation could lead to enhanced performance in routing efficiency and quality.

## 6 Acknowledgements

We would like to thank the National Key Research and Development Program of China (2020YFA0907000), the National Natural Science Foundation of China (32370657, 32271297, 82130055, 62072435), and the Major Key Project of PCL (No. PCL2023A03) for providing financial supports for this study and publication charges. The numerical calculations in this study were supported by ICT Computer X center, CAS Xiandao-1 and Pengcheng Cloudbrain.

## References

[1] Alpert, C. J. (1998). The ispd98 circuit benchmark suite. In *Proceedings of the 1998 international symposium on Physical design*, pages 80–85.

[2] Chen, H.-Y. and Chang, Y.-W. (2009). Global and detailed routing. In *Electronic Design Automation*, pages 687–749. Elsevier.

[3] Chen, T., Xiong, S., He, H., and Yu, B. (2023). Trouter: Thermal-driven pcb routing via non-local crisscross attention networks. *IEEE Transactions on Computer-Aided Design of Integrated Circuits and Systems*.

[4] Cheng, R., Lyu, X., Li, Y., Ye, J., Hao, J., and Yan, J. (2022). The policy-gradient placement and generative routing neural networks for chip design. *Advances in Neural Information Processing Systems*, 35:26350–26362.

[5] Cheng, R. and Yan, J. (2021). On joint learning for solving placement and routing in chip design. *Advances in Neural Information Processing Systems*, 34:16508–16519.

[6] Cho, M., Lu, K., Yuan, K., and Pan, D. Z. (2007). Boxrouter 2.0: Architecture and implementation of a hybrid and robust global router. In *2007 IEEE/ACM International Conference on Computer-Aided Design*, pages 503–508. IEEE.

[7] Chu, C. and Wong, Y.-C. (2005). Fast and accurate rectilinear steiner minimal tree algorithm for vlsi design. In *Proceedings of the 2005 international symposium on Physical design*, pages 28–35.

[8] Du, X., Wang, C., Zhong, R., and Yan, J. (2023). Hubrouter: Learning global routing via hub generation and pin-hub connection. In *Thirty-seventh Conference on Neural Information Processing Systems*.

[9] Ganley, J. L. and Cohoon, J. P. (1994). Routing a multi-terminal critical net: Steiner tree construction in the presence of obstacles. In *Proceedings of IEEE International Symposium on Circuits and Systems-ISCAS'94*, volume 1, pages 113–116. IEEE.

[10] Garey, M. R. and Johnson, D. S. (1977). The rectilinear steiner tree problem is np-complete. *SIAM Journal on Applied Mathematics*, 32(4):826–834.

[11] Hanan, M. (1966). On steiner's problem with rectilinear distance. *SIAM Journal on Applied mathematics*, 14(2):255–265.

[12] He, K., Zhang, X., Ren, S., and Sun, J. (2016). Deep residual learning for image recognition. In *Proceedings of the IEEE conference on computer vision and pattern recognition*, pages 770–778.

[13] Ho, J.-M., Vijayan, G., and Wong, C.-K. (1990). New algorithms for the rectilinear steiner tree problem. *IEEE transactions on computer-aided design of integrated circuits and systems*, 9(2):185–193.

[14] Huang, Z., Wang, X., Huang, L., Huang, C., Wei, Y., and Liu, W. (2019). Ccnet: Criss-cross attention for semantic segmentation. In *Proceedings of the IEEE/CVF international conference on computer vision*, pages 603–612.

[15] Juhl, D., Warme, D. M., Winter, P., and Zachariasen, M. (2018). The geosteiner software package for computing steiner trees in the plane: an updated computational study. *Mathematical Programming Computation*, 10:487–532.

[16] Kastner, R., Bozorgzadeh, E., and Sarrafzadeh, M. (2002). Pattern routing: Use and theory for increasing predictability and avoiding coupling. *IEEE Transactions on Computer-Aided Design of Integrated Circuits and Systems*, 21(7):777–790.

[17] Kramer, M. and Van Leeuwen, J. (1984). The complexity ofwirerouting and finding minimum area layouts for arbitrary vlsicircuits. *Adv. Comput. Res*, 2:129–146.

[18] Lai, Y., Liu, J., Tang, Z., Wang, B., Hao, J., and Luo, P. (2023). Chipformer: Transferable chip placement via offline decision transformer. *arXiv preprint arXiv:2306.14744*.

[19] Lai, Y., Mu, Y., and Luo, P. (2022). Maskplace: Fast chip placement via reinforced visual representation learning. *Advances in Neural Information Processing Systems*, 35:24019–24030.

[20] Liao, H., Zhang, W., Dong, X., Poczos, B., Shimada, K., and Burak Kara, L. (2020). A deep reinforcement learning approach for global routing. *Journal of Mechanical Design*, 142(6):061701.

[21] Lin, T.-Y., Goyal, P., Girshick, R., He, K., and Dollár, P. (2017). Focal loss for dense object detection. In *Proceedings of the IEEE international conference on computer vision*, pages 2980–2988.

[22] Liu, C.-H., Kuo, S.-Y., Lee, D., Lin, C.-S., Weng, J.-H., and Yuan, S.-Y. (2012). Obstacle-avoiding rectilinear steiner tree construction: A steiner-point-based algorithm. *IEEE Transactions on Computer-Aided Design of Integrated Circuits and Systems*, 31(7):1050–1060.

[23] Liu, J., Chen, G., and Young, E. F. (2021). Rest: Constructing rectilinear steiner minimum tree via reinforcement learning. In *2021 58th ACM/IEEE Design Automation Conference (DAC)*, pages 1135–1140. IEEE.

[24] Liu, J., Pui, C.-W., Wang, F., and Young, E. F. (2020). Cugr: Detailed-routability-driven 3d global routing with probabilistic resource model. In *2020 57th ACM/IEEE Design Automation Conference (DAC)*, pages 1–6. IEEE.

[25] Liu, J. and Young, E. F. (2023). Edge: Efficient dag-based global routing engine. In *2023 60th ACM/IEEE Design Automation Conference (DAC)*, pages 1–6. IEEE.

[26] Liu, S., Pu, Y., Liao, P., Wu, H., Zhang, R., Chen, Z., Lv, W., Lin, Y., and Yu, B. (2022). Fastgr: Global routing on cpu-gpu with heterogeneous task graph scheduler. *IEEE Transactions on Computer-Aided Design of Integrated Circuits and Systems*.

[27] Liu, W.-H., Kao, W.-C., Li, Y.-L., and Chao, K.-Y. (2013). Nctu-gr 2.0: Multithreaded collision-aware global routing with bounded-length maze routing. *IEEE Transactions on computer-aided design of integrated circuits and systems*, 32(5):709–722.

[28] Mahboubi, S., Ninomiya, H., Kamio, T., Asai, H., et al. (2021). A nesterov's accelerated quasi-newton method for global routing using deep reinforcement learning. *Nonlinear Theory and Its Applications, IEICE*, 12(3):323–335.

[29] Mirhoseini, A., Goldie, A., Yazgan, M., Jiang, J. W., Songhori, E., Wang, S., Lee, Y.-J., Johnson, E., Pathak, O., Nazi, A., et al. (2021). A graph placement methodology for fast chip design. *Nature*, 594(7862):207–212.

[30] Nam, G.-J., Yildiz, M., Pan, D. Z., and Madden, P. H. (2007). Ispd placement contest updates and ispd 2007 global routing contest. In *Proceedings of the 2007 international symposium on Physical design*, pages 167–167.

[31] Ronneberger, O., Fischer, P., and Brox, T. (2015). U-net: Convolutional networks for biomedical image segmentation. In *Medical Image Computing and Computer-Assisted Intervention–MICCAI 2015: 18th International Conference, Munich, Germany, October 5-9, 2015, Proceedings, Part III 18*, pages 234–241. Springer.

[32] Utyamishev, D. and Partin-Vaisband, I. (2020). Late breaking results: A neural network that routes ics. In *2020 57th ACM/IEEE Design Automation Conference (DAC)*, pages 1–2. IEEE.

[33] Wu, T.-H., Davoodi, A., and Linderoth, J. T. (2010). Grip: Global routing via integer programming. *IEEE Transactions on Computer-Aided Design of Integrated Circuits and Systems*, 30(1):72–84.

[34] Xie, Z., Huang, Y.-H., Fang, G.-Q., Ren, H., Fang, S.-Y., Chen, Y., and Hu, J. (2018). Routenet: Routability prediction for mixed-size designs using convolutional neural network. In *2018 IEEE/ACM International Conference on Computer-Aided Design (ICCAD)*, pages 1–8. IEEE.

[35] Zhou, Z., Zhu, Z., Chen, J., Ma, Y., Yu, B., Ho, T.-Y., Lemieux, G., and Ivanov, A. (2019). Congestion-aware global routing using deep convolutional generative adversarial networks. In *2019 ACM/IEEE 1st Workshop on Machine Learning for CAD (MLCAD)*, pages 1–6. IEEE.

# A  Related Information

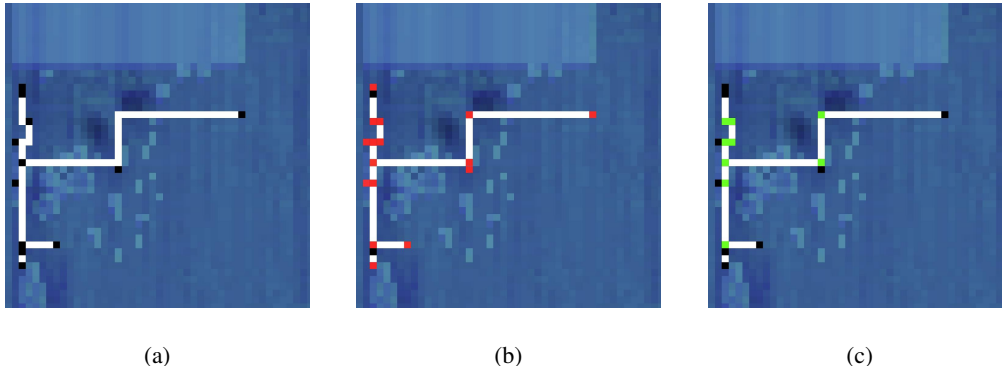

<center>(a)             (b)             (c)</center>

Figure S1: **Differences between hub points and candidate points.** (a) Routing result (white lines) generated by traditional global router for a 12-pin (black squares) net . (b) The hub points (red squares) extracted by [8] as their label points. (c) The candidate points (green squares) labeled by NeuralSteiner as the learning targets.

Table S1: **Summary of predicted candidate points for ISPD07.** We respectively show the scale size, number of nets, average / maximum number of pins and candidate points added by NeuralSteiner.

| Case | adaptec01_2d | adaptec02_2d | adaptec03_2d | adaptec04_2d | adaptec05_2d | newblue01_2d | newblue02_2d | newblue03_2d |
|---|---|---|---|---|---|---|---|---|
| Size | 324×324 | 424×424 | 774×779 | 774×779 | 465×468 | 399×399 | 557×463 | 973×1256 |
| Cap. (V/H) | 70 / 70 | 80 / 80 | 62 / 62 | 62 / 62 | 110 / 110 | 62 / 62 | 110 /110 | 80 /80 |
| #Nets | 219794 | 260159 | 466295 | 515304 | 867441 | 331663 | 463213 | 551667 |
| Avg. #Pins | 4.29 | 4.09 | 4.02 | 3.71 | 4.03 | 3.73 | 3.83 | 3.50 |
| Max #Pins | 2271 | 1935 | 3713 | 3974 | 9863 | 12335 | 8089 | 16008 |
| #Pins $\leq 10$ | 92.90% | 94.03% | 95.24% | 96.09% | 94.94% | 95.50% | 96.02% | 97.12% |
| Avg. #Cands | 4.61 | 4.41 | 4.76 | 4.34 | 4.31 | 3.86 | 3.88 | 3.56 |
| Max #Cands | 2474 | 2145 | 4189 | 4226 | 10681 | 13109 | 8517 | 16951 |

# B  Implementation details

## B.1  NeuralSteiner Network Architecture.

We employ a modified ResNet-34 structure [12] as the backbone of our model and combined it with the recurrent crisscross attention mechanism (RCCA) [14]. Our network accepts two channels, pin map and overflow map, as input, while the output is a single-channel map indicating the positions of the predicted candidate points. To retain more accurate spatial location details of feature maps for the construction of RST, we remove the max-pooling layer from the ResNet backbone. Every two residual blocks are grouped in one pair. Thanks to the $3 \times 3$ convolution, the number of channels gradually increasing from 2 to 256. After 4 pairs of residual blocks, a convolutional layer is applied to obtain the feature map H of dimension reduction. Then, H is fed into the RCCA module to generate a new feature map H0 which aggregate non-local contextual information. In RCCA module, we sequentially apply two crisscross attention modules and assign different weights (attention map) to each part of the feature. After the RCCA, we concatenate the feature H0 processed by RCCA with the local and dimension-reduced feature X. It is followed by another 4 pairs of residual blocks with $3 \times 3$ convolution and the number of channels gradually decreasing from 256 to 1. The output feature map undergoes a sigmoid operation to obtain the final predicted candidate point map.

The crisscross attention (CCA) mechanism is adopted in NeuralSteiner to capture long-range dependencies in feature maps and aggregate full-net spatial and overflow information by computing attention along both horizontal and vertical directions for each pixel, which is shown in Fig.2. Given an input feature map $X \in \mathbb{R}^{C \times W \times H}$, where $C$ is the number of channels, $W$ and $H$ is the width and height respectively, the CCA starts by computing queries, keys, and values using $1 \times 1$ convolutions

with weight matrices $W_q$, $W_k$, and $W_v$:

$$Q = W_q * X, \quad K = W_k * X, \quad V = W_v * X$$

Here, $*$ denotes the convolution operation and where $\{Q, K\} \in \mathbb{R}^{C' \times W \times H}$ and $V \in \mathbb{R}^{C \times W \times H}$. For each pixel $(i, j)$ in the input feature map $X$, we use $\mathbb{K}_{(i,j)}$ to denote the set of feature vectors extracted from $K$ which are in the same row or column with position $(i, j)$ and $\mathbb{K}_{(i,j)} \in \mathbb{R}^{(W+H-1) \times C'}$. first compute the horizontal attention weights $A_{i,j}^H$ by applying the softmax function to the dot product of the query at row $i$ and the key at position $(i, j)$:

$$d_{(i,j),k} = Q_{(i,j)} \cdot \mathbb{K}_{(i,j),k}^T$$

where $d_{(i,j),k} \in D$ is the degree of correlation between $Q_{(i,j)}$ and $\mathbb{K}_{(i,j),k}$, $k = [1, ..., W + H - 1]$. Note that $D \in \mathbb{R}^{(H+W-1) \times (W \times H)}$, we compute the attention map $A$ by applying the softmax function to $D$. We also use $\mathbb{V}_{(i,j)}$ to denote the collection of feature vectors extracted from $V$ which are in the same row or column with position $(i, j)$, then the crisscross spatial and overflow information can be aggregated by:

$$Y_{(i,j)} = \sum_{k=0}^{W+H-1} A_{(i,j),k} \mathbb{V}_{(i,j),k} + X_{(i,j)}$$

The output feature map $Y \in \mathbb{R}^{C \times H \times W}$ is then returned as the result of the crisscross attention mechanism. As shown in Fig. 3, after two CCA operations, the information of positions in different rows and columns from the coordinates $(i, j)$ can also be aggregated, thus enabling learning of long-range spatial and overflow correlation.

## B.2 Training details for candidate point prediction phase.

We adopt CUGR [24] to conduct routing for dataset construction, and we choose bigblue04_3d, newblue03_3d, newblue04_3d and newblue07_3d (name of the cases in ISPD-07) as the training cases and generate 25K training samples using each case, which is more than that in HubRouter [8]. Thus, we have a total of nearly 100K for training.

We use learning rate in [0.001, 0.00001] and reduce the learning rates by 0.5 if the validation loss does not decrease in 2 epochs. The training will continue until the validation loss no longer decreases for over 10 epochs or the number of epoch reaches a maximum of 100. In the training loss, we use $c_{fl} = 1.0$, $c_{di} = 1.0$ and $c_{of} = 2.0$ to encourage the exploration of lower-overflow candidate points. The number of ResNet blocks is 8 before the RCCA module and 8 behind it.

Each experiment in this work is conducted on a system equipped with an Intel(R) Xeon(R) Gold 6230R CPU, NVIDIA A800 (80 GB) GPU, and 250 GB RAM. We repeat 3 times under different seeds for HubRouter on the small nets test set and ISPD98 and report the error bars, and then choose the seed with best overflow for HubRouter (GAN) to conduct the ISPD07 experiment.

---

**Algorithm 1** Training in candidate point prediction phase

---

**Require:** $num\_iters$ (number of training iterations), $B$ (minibatch size), $trainingSet$ (training set), $c_{fl}$, $c_{di}$, $c_{of}$ (loss function coefficients)
**Ensure:** $\theta$ (model parameters)
1: Initialize model parameters $\theta$
2: **for** $iter = 1$ to $num\_iters$ **do**
3:      Sample a batch $\mathcal{B} \subset trainingSet$ with $|\mathcal{B}| = B$
4:      Pad all pin maps, overflow maps and label point maps in $\mathcal{B}$ to the same size
5:      Descend the stochastic gradient of loss:

$$\nabla_\theta \left[ c_{fl} \cdot \ell_{focal} + c_{di} \cdot \ell_{dice} + c_{of} \cdot \ell_{of} \right]$$

6: **end for**
7: **return** Model with parameters $\theta$

---

## B.3 Parallel Routing Tasks Construction.

---

**Algorithm 2** Parallel routing tasks construction

---

1: **Input:** A set of nets $\mathbb{N}$ need to route in a chip design where each net has a corresponding HPWL. A set of net tasks $\mathbb{T}$.
2: $\mathbb{N} \leftarrow \{net_1, net_2, \ldots, net_n\}$
3: $\mathbb{T} \leftarrow \{\}$
4: $\mathbb{N}_{sorted} \leftarrow \text{Sort}(\mathbb{N}, \text{by } HPWL)$           ▷ Sort nets by HPWL
5: **for** each $net_i \in \mathbb{N}_{sorted}$ **do**
6:     $assigned \leftarrow \text{False}$
7:     **for** each $task_j \in \mathbb{T}$ **do**
8:         **if** $net_i$ has no conflicts with all $net$ in $task_j$ **then**
9:             $task_j \leftarrow task_j \cup \{net_i\}$
10:             $assigned \leftarrow \text{True}$
11:             **break**
12:         **end if**
13:     **end for**
14:     **if not** $assigned$ **then**
15:         $newTask \leftarrow \{\}$
16:         $newTask \leftarrow newTask \cup \{net_i\}$
17:         $\mathbb{T} \leftarrow \mathbb{T} \cup \{newTask\}$
18:     **end if**
19: **end for**
20: **return** $\mathbb{T}$

---

## B.4 Net Augmented Graph Construction.

---

**Algorithm 3** Net augmented graph construction

---

**Require:** Candidate point map $C$ (2D map), pin map $P$ (2D map), overflow map $O$ (2D map).
**Ensure:** $netAugmentedGraph$ (Graph).
1: $mergedMap \leftarrow C + P$
2: $points \leftarrow \{(x, y) \mid mergedMap[x][y] \geq 1\}$
3: $NAG \leftarrow$ empty graph
4: **for all** $p \in points$ **do**
5:     Add $p$ as a node in $NAG$
6: **end for**
7: **for all** $(p_i, p_j) \in \text{pairs}(points)$ **do**
8:     **if** $p_i.x = p_j.x$ **or** $p_i.y = p_j.y$ **then**
9:         **if** No other points on the line between $p_i$ and $p_j$ **then**
10:             $weight \leftarrow \text{calculate\_weight}(p_i, p_j)$
11:             Add edge $(p_i, p_j, weight)$ to $netAugmentedGraph$
12:         **end if**
13:     **end if**
14: **end for**
15: **while** number\_of\_connected\_components($netAugmentedGraph$) $> 1$ **do**
16:     Select one point from each connected component
17:     Add edges between selected points
18: **end while**
19: **return** $netAugmentedGraph$

---

## B.5 Overflow-avoiding RST Construction.

---

**Algorithm 4** Overflow-Avoiding RST Construction

---

**Require:** $NAG$ (Graph), $Pins$ (Set of pins)
**Ensure:** $RST$ (Constructed RST)
1: Components $\leftarrow \{\{p\} \mid p \in \text{Pins}\}$
2: Parallelize computation of dist$(C_i, C_j)$ for $C_i, C_j \in$ Components on $NAG$
3: **while** $|\text{Components}| > 1$ **do**
4:     $(C_1, C_2) \leftarrow \arg\min_{C_i, C_j \in \text{Components}} \text{dist}(C_i, C_j)$
5:     RST $\leftarrow$ RST $\cup \{(C_1, C_2)\}$
6:     $C_{\text{new}} \leftarrow C_1 \cup C_2$
7:     Components $\leftarrow$ (Components $\setminus \{C_1, C_2\}$)
8:     Delete dist$(C_1, C_k)$ and dist$(C_2, C_k)$ for all $C_k \in$ Components
9:     Update dist$(C_{\text{new}}, C_k)$ for all $C_k \in$ Components in parallel
10:     Components $\leftarrow$ Components $\cup \{C_{\text{new}}\}$
11: **end while**
12: **return** RST

---

**Time complexity and scalability of the RST construction algorithm.** We use Dijkstra's algorithm to search for the shortest path between every two points. Suppose the number of pin in one net is $N_{pin}$. Given that NAG is a sparse graph, and the number of edges and nodes in the NAG is $(O(N_{pin}))$ according to Table S1, Dijkstra's algorithm with a binary heap yields a time complexity of $(O(N_{pin} \log N_{pin}))$. The total time complexity is $(O((N_{pin})^2 \log N_{pin}))$ to calculate the shortest path for all points in the connected component. In each iteration, we connect the two components with the shortest distance and update the distances matrix. It will end with a connected component containing all pins. So, in the worst case, the total time complexity of the RST construction algorithm is $(O((N_{pin})^4 \log N_{pin}))$. Table S1 shows that the pin size for most of nets in dataset is no more than 10, indicating that the algorithm will be efficient.

Additionally, the algorithm can be accelerated by simple heuristic rules. When calculating the distance from one connected component to another, we compute and filter out a fixed number of node pairs with the shortest Euclidean distance, performing Dijkstra's algorithm only on these limited node pairs. This reduces the complexity of the RST construction algorithm to $(O((N_{pin})^3 \log N_{pin}))$. As shown in Table S3, the accelerated algorithm NeuralSteiner-Ltd significantly improves solving efficiency while achieving similar wirelength and overflow compared to the original version.

---

**Algorithm 5** Inference Algorithm

---

**Require:** $netlist$ (netlist), $resources$ (routing resources)
**Ensure:** $routes$ (routing results)
1: Initialize $routes \leftarrow \emptyset$
2: // Parallel routing tasks construction
3: $T \leftarrow$ ParallelRoutingTasksConstruction$(netlist)$         $\triangleright$ Using Algorithm 1
4: **for** each task $t$ in $T$ **do**
5:     // NeuralSteiner for candidate point prediction
6:     $candidatePointMaps \leftarrow$ NeuralSteiner$(t)$         $\triangleright$ Predict candidate point maps
7:     // Net augmented graph construction
8:     $G \leftarrow$ NetAugmentedGraphConstruction$(t, candidatePointMaps)$   $\triangleright$ Using Algorithm 3
9:     // Overflow-Avoiding RST Construction
10:     $routingResults \leftarrow$ OverflowAvoidingRSTConstruction$(G)$     $\triangleright$ Using Algorithm 2
11:     Add $routingResults$ to $routes$
12: **end for**
13: **return** $routes$

---

# C Additional Results

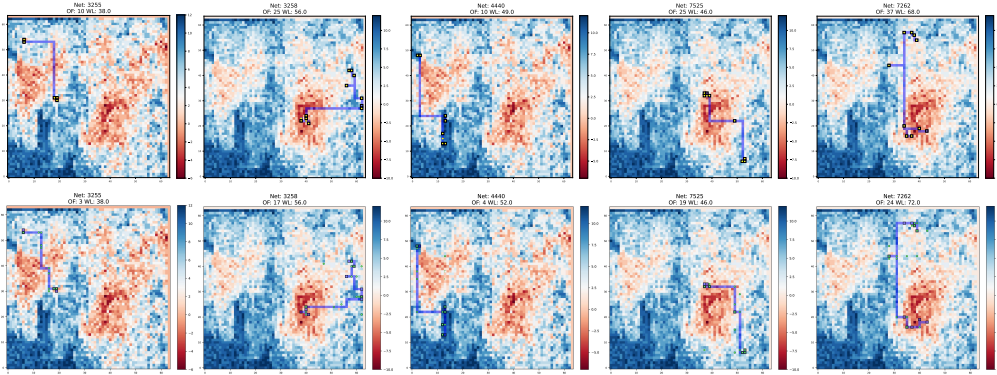

Figure S2: **Comparison of actual solutions of Hubrouter (first line) and our NeuralSteiner (second line), randomly sampled from ibm01.** Pins (black hollow square), hub points (yellow circle) candidate points (green circle) and the routes (purple line) are shown in figures.

Table S2: **Experiments on 4 kinds of small nets.** We train three architectures of HubRouter with default settings and test the results of three different random seeds for each HubRouter model. NeuralSteiner is tested three times to obtain the mean and standard deviation values of generation time. The best results are highlighted in bold.

| Metric | Case | HubRouter (VAE) | HubRouter (DPM) | HubRouter (GAN) | NeuralSteiner |
|---|---|---|---|---|---|
| Correctness Rate (%) | Route-small-4 | **1.000 ± 0.000** | **1.000 ± 0.000** | **1.000 ± 0.000** | **1.000** |
| | Route-small | **1.000 ± 0.000** | **1.000 ± 0.000** | **1.000 ± 0.000** | **1.000** |
| | Route-large-4 | **1.000 ± 0.000** | **1.000 ± 0.000** | **1.000 ± 0.000** | **1.000** |
| | Route-large | **1.000 ± 0.000** | **1.000 ± 0.000** | **1.000 ± 0.000** | **1.000** |
| Wirelength Rate (%) | Route-small-4 | 1.086 ± 0.031 | 1.072 ± 0.020 | 1.012 ± 0.011 | **1.003** |
| | Route-small | 1.051 ± 0.017 | 1.191 ± 0.006 | **1.002 ± 0.001** | 1.003 |
| | Route-large-4 | 1.124 ± 0.026 | 1.102 ± 0.039 | 1.004 ± 0.021 | **1.002** |
| | Route-large | 1.045 ± 0.023 | 1.264 ± 0.014 | **1.001 ± 0.002** | 1.002 |
| Generation Time (GPU Sec) | Route-small-4 | 5.33 ± 0.09 | 514.71 ± 4.42 | **5.14 ± 0.08** | 6.60 |
| | Route-small | **5.68 ± 0.14** | 506.20 ± 2.99 | 6.63 ± 0.35 | 7.01 |
| | Route-large-4 | **5.41 ± 0.15** | 511.42 ± 2.85 | 5.68 ± 0.16 | 6.89 |
| | Route-large | 7.17 ± 0.05 | 531.98 ± 3.63 | **7.16 ± 0.21** | 8.77 |

Table S3: **Comparison of NeuralSteiner with limited connected component nodes (NeuralSteiner-Ltd) and original NeuralSteiner on ISPD-98 (ibm01-06).**

| Metric | Model | ibm01 | ibm02 | ibm03 | ibm04 | ibm05 | ibm06 |
|---|---|---|---|---|---|---|---|
| WL | NeuralSteiner | 61735 | **170405** | **148036** | 166648 | **415684** | 283727 |
| | NeuralSteiner-Ltd | **61654** | 170430 | 148730 | **166160** | 415872 | **283215** |
| OF | NeuralSteiner | **2033** | **3953** | 1754 | **2794** | 18 | **3042** |
| | NeuralSteiner-Ltd | 2042 | 4046 | **1743** | 2883 | 18 | 3095 |
| Time (Sec) | NeuralSteiner | 24.75 | 33.18 | 41.94 | 46.63 | 67.52 | 63.44 |
| | NeuralSteiner-Ltd | **13.68** | **31.84** | **21.20** | **23.30** | **47.65** | **36.26** |

